# Structural and Behavioral Evolution
# of Recurrent Networks

**Gregory M. Saunders, Peter J. Angeline, and Jordan B. Pollack**
Laboratory for Artificial Intelligence Research
Department of Computer and Information Science
The Ohio State University
Columbus, Ohio 43210
saunders@cis.ohio-state.edu

## Abstract

This paper introduces GNARL, an evolutionary program which induces recurrent neural networks that are structurally unconstrained. In contrast to constructive and destructive algorithms, GNARL employs a population of networks and uses a fitness function's unsupervised feedback to guide search through network space. Annealing is used in generating both gaussian weight changes and structural modifications. Applying GNARL to a complex search and collection task demonstrates that the system is capable of inducing networks with complex internal dynamics.

## 1 INTRODUCTION

A variety of methods to induce network architecture exist. Some start with a very simple network and incrementally add nodes and links (Hanson 1990; Fahlman & Lebiere, 1990; Fahlman 1991; Chen, et al., 1993); others start with a large network and then prune off superfluous pieces (Mozer & Smolensky, 1989; Cun, Denker, and Solla, 1990; Hassibi & Stork, 1993; Omlin & Giles, 1993). But these *constructive* and *destructive* algorithms are monotonic extremes that ignore a more moderate solution: "dynamically add or remove pieces of architecture as needed." Moreover, by exclusively exploring either feedforward networks (e.g., Ash, 1989), fully-connected recurrent networks (e.g., Chen, et al. 1993), or some restricted middle ground (e.g., Fahlman, 1991), these algorithms allow only limited structural change. Finally, constructive and destructive algorithms are supervised methods

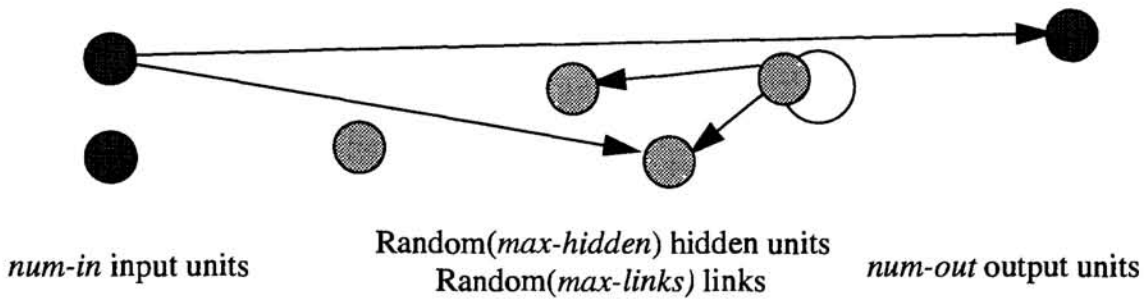

num-in input units

Random(*max-hidden*) hidden units
Random(*max-links)* links

num-out output units

Figure 1:  Sample initial network. The number of input nodes and number of output nodes is fixed for the particular task, but the number of hidden units and the connectivity (although bounded), is random.

which rely on complex predicates to determine when to add or delete pieces of network architecture (e.g., "when rate of improvement falls below threshold").

Genetic algorithms (Holland 1975), on the other hand, are unsupervised methods which can induce networks by making stochastic modifications to a population of bitstrings, each of which is interpreted as a network. Most studies, however, still assume a fixed structure for the network (e.g., Belew et al., 1990; Jefferson, et al., 1991; see also Schaffer, et al. 1992), and those that do not allow only limited structural change (e.g., Potter, 1992, and Karu-nanithi et al., 1992).

*Evolutionary programming* (Fogel, 1992) is an alternate optimization technique which, when applied to network induction, obviates the need for a bitstring-to-network mapping by mutating networks directly. Furthermore, because EP does not employ crossover (an operator of questionable efficacy on distributed representations), it is a better candidate for inducing network structures (Angeline, Saunders, and Pollack, 1993; Fogel et al., 1990).

## 2  THE GNARL ALGORITHM

GNARL *(GeNeralized Acquisition of Recurrent Links)* is an evolutionary program that non-monotonically constructs recurrent networks to solve a given task. It begins with an initial population of *n* random individuals; a sample network *N* is shown in Figure 1. The number of input nodes *(num-in)* and number of output nodes *(num-out)* are fixed for a given task; the number of hidden nodes as well as the connections among them are free to vary. Self-links as well as general loops are allowed. Thus GNARL's search space is $N \equiv \{N$: network *N* has *num-in* input nodes and *num-out* output nodes$\}$.

In each epoch of search, the networks are ranked by a user-supplied fitness function $f: N \rightarrow \mathbf{R}$, where $\mathbf{R}$ represents the reals. Reproduction of the best $^n/_2$ individuals entails modifying both the weights and structure of each parent network *N*. First, the temperature $T(N)$ is calculated:

$$T(N) = 1 - \frac{f(N)}{f_{max}} \tag{1}$$

where $f_{max}$ (provided by the user) is the maximum possible fitness for a given task. This

measure of $N$'s performance is used to anneal the *structural* and *parametric* (Barto, 1990) similarity between parent and offspring, so that networks with a high temperature are mutated severely, and those with a low temperature are mutated only slightly. This allows a coarse-grained search initially, and a finer-grained search as a network approaches a solution (cf. Kirkpatrick et al., 1983).

More concretely, parametric mutations are accomplished by perturbing each weight with gaussian noise, whose variance is $T(N)^2$:

$$w \leftarrow w + \text{Normal } (0; T(N)), \quad \forall w \in N \tag{2}$$

Structural mutations are accomplished by:

- adding   $k_1$ hidden nodes with probability $p_{add\text{-}node}$
- deleting $k_2$ hidden nodes with probability $p_{delete\text{-}node}$
- adding   $k_3$ links with probability $p_{add\text{-}link}$
- deleting $k_4$ links with probability $p_{delete\text{-}link}$

where each $k_i$ is selected uniformly from a user-defined range, again annealed by $T(N)$. When a node is added, it is initialized without connections; when a node is deleted, all its incident links are removed. All new links are initialized to 0. (See also Angeline, Saunders, and Pollack, 1993.)

## 3   RESULTS

GNARL was tested on a simple control task – the *Tracker* task of Jefferson, et al. (1991) and Koza (1992). In this problem, a simulated ant is placed on a two-dimensional toroidal grid and must maximize the number of pieces of food it collects in a given time period (Figure 2a). Each ant is controlled by a network with two input nodes and four output nodes (Figure 2b). At each step, the action whose corresponding output node has maximum activation is performed. Fitness is the number of grid positions cleared within 200 time steps.

The experiments used a population of 100 networks. In the first run (2090 generations), GNARL found a network (Figure 3b) that cleared 81 grid positions within the 200 time steps. Figure 4 shows the state of the output units of the network over three different sets of inputs. Each point is a triple of the form (*move, right, left*). (No-op is not shown because it was never used in the final network.) Figure 4a shows the result of supplying to the network 200 "food" inputs – a fixed point that executes "Move." Figure 4b shows the sequence of states reached when 200 "no food" signals are supplied to the network – a collection of points describing a limit cycle of length 5 that repeatedly executes the sequence "Right, Right, Right, Right, Move." These two attractors determine the response of the network to the task (Figure 4c,d); the additional points in Figure 4c are transients encountered as the network alternates between these attractors.

However, not all evolved network behaviors are so simple as to approximate an FSA (Pollack, 1991). In a second run (1595 generations) GNARL induced a network that cleared 82 grid points within the 200 time steps. Figure 5 demonstrates the behavior of this network. Once again, the "food" attractor, shown in Figure 5a, is a single point in the space that always executes "Move." The "no food" behavior, however, is not an FSA; instead, it is a

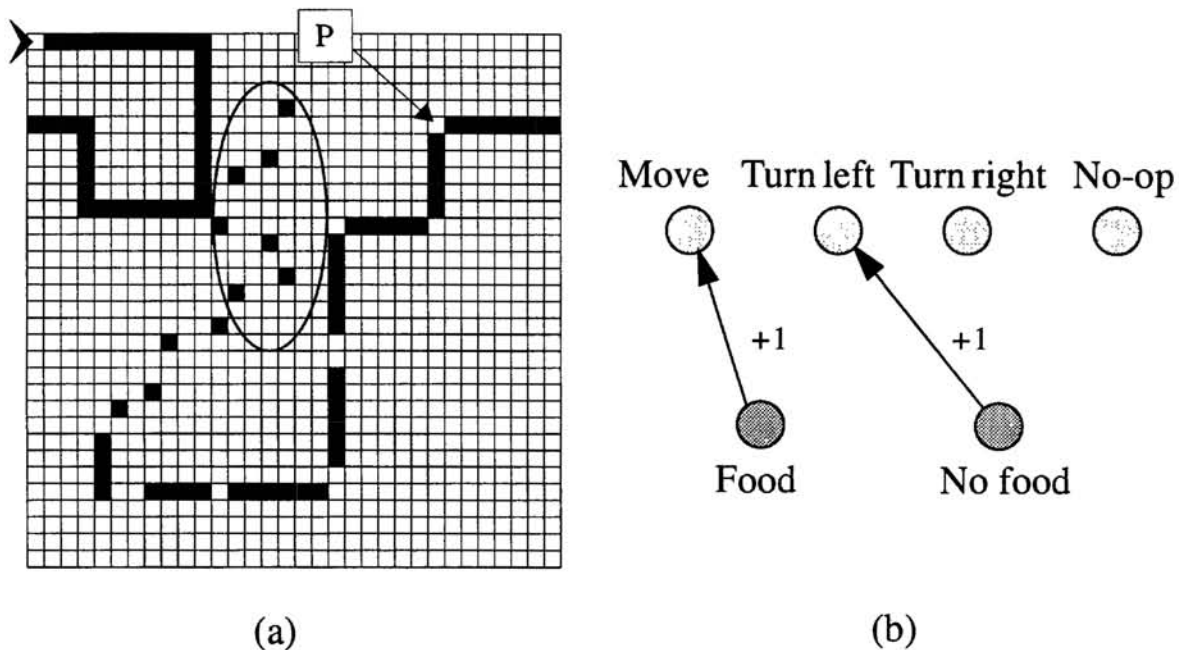

(a)                                          (b)

Figure 2: The ant problem. (a) The trail is connected initially, but becomes progressively more difficult to follow. The underlying 2-d grid is toroidal, so that position "P" is the first break in the trail. The ellipse indicates the 7 pieces of food that the network of the second run failed to reach. (b) The semantics of the I/O units for the ant network. The first input node denotes the presence of food in the square directly in front of the ant; the second denotes the absence of food in this same square. No-op, from Jefferson, allows the network to stay in one position while activation flows through recurrent links. This particular network "eats" 42 pieces of food before spinning endlessly in place at position P, illustrating a very deep local minimum in the search space.

quasiperiodic trajectory of points shaped like a "D" in output space (Figure 5b). The placement of the "D" is in the "Move / Right" corner of the space and encodes a complex alternation between these two operations (Figure 5d).

## 4  CONCLUSIONS

Artificial architectural constraints (such as "feedforwardness") close the door on entire classes of behavior; forced liberties (such as assumed full recurrence) may unnecessarily increase structural complexity or learning time. By relying on a simple stochastic process, GNARL strikes a middle ground between these two, allowing the network's complexity and behavior to emerge in response to the demands of the task.

### Acknowledgments

The research reported in this paper has been partially supported by Office of Naval Research grants N00014-93-1-0059 and N00014-92-J-1195. We are indebted to all those who read and reviewed this work, especially John Kolen, Ed Large, and Barbara Becker.

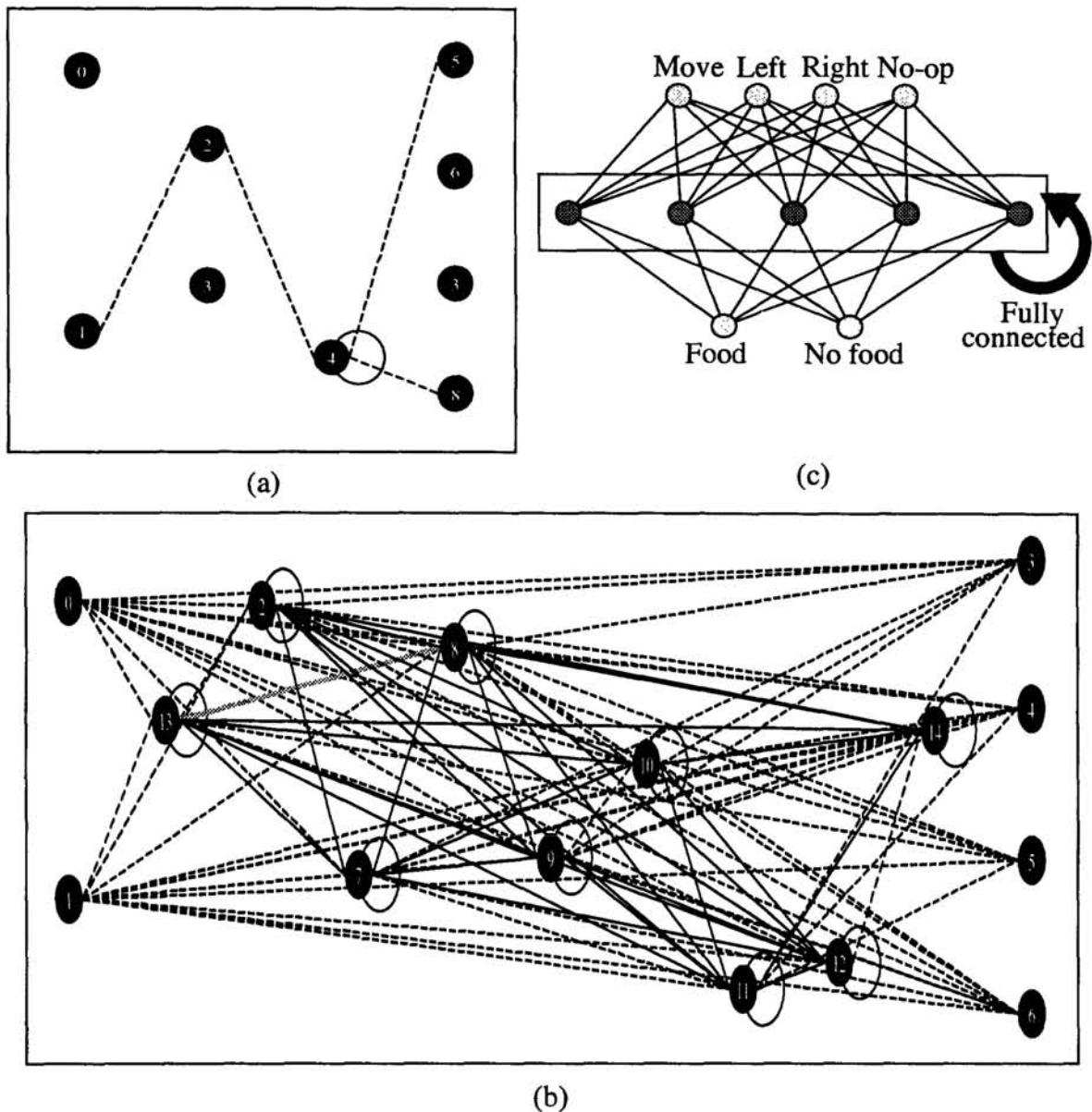

(a)

(c)

(b)

Figure 3: The Tracker Task, first run. (a) The best network in the initial population. Nodes 0 & 1 are input, nodes 5-8 are output, and nodes 2-4 are hidden nodes. (b) Network induced by GNARL after 2090 generations. Forward links are dashed; bidirectional links & loops are solid. The light gray connection between nodes 8 and 13 is the sole backlink. This network clears the trail in 319 epochs. (c) Jefferson et al.'s fixed network structure for the Tracker task.

## References

Angeline, P., Saunders, G., Pollack, J. (1993). An evolutionary algorithm that constructs recurrent neural networks. LAIR Technical Report 93-PA-GNARL, The Ohio State University, Columbus Ohio. To be published in *IEEE Transactions on Neural Networks*.

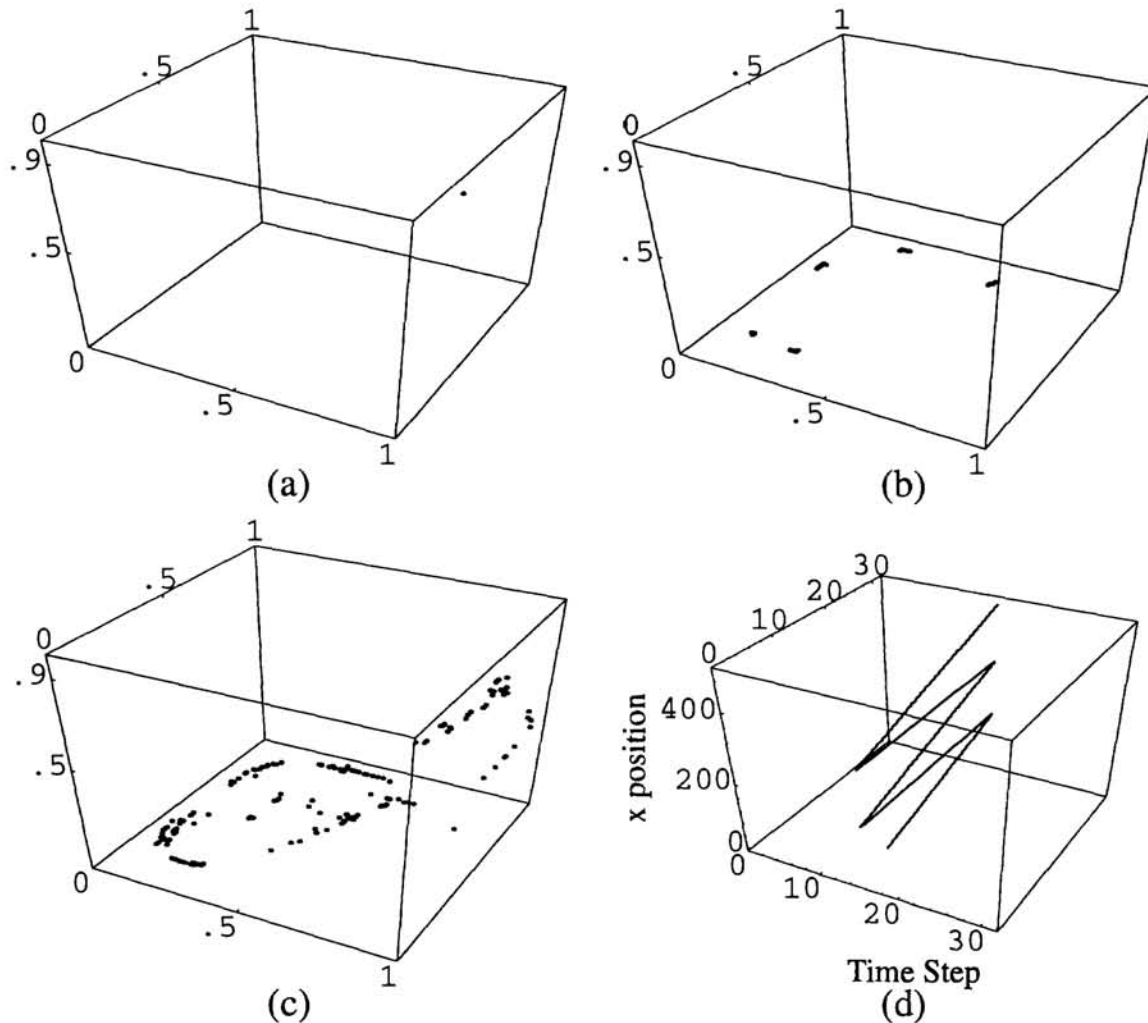

Figure 4: Limit behavior of the network that clears the trail in 319 steps. Graphs show the state of the output units Move, Right, Left. (a) Fixed point attractor that results for sequence of 200 "food" signals; (b) Limit cycle attractor that results when a sequence of 200 "no food" signals is given to network; (c) All states visited while traversing the trail; (d) The x position of the ant over time when run on an empty grid.

Ash, T. (1989). "Dynamic node creation in backpropagation networks," *Connection Science*, 1:365–375.

Barto, A. G. (1990). Connectionist learning for control. In Miller, W. T. III, Sutton, R. S., and Werbos, P. J., editors, *Neural Networks for Control*. Chapter 1, pages 5-58. MIT Press, Cambridge.

Belew, R. K., McInerney, J., and Schraudolf, N. N. (1990). Evolving networks: Using the genetic algorithm with connectionist learning. Technical Report CS90-174, University of California, San Diego.

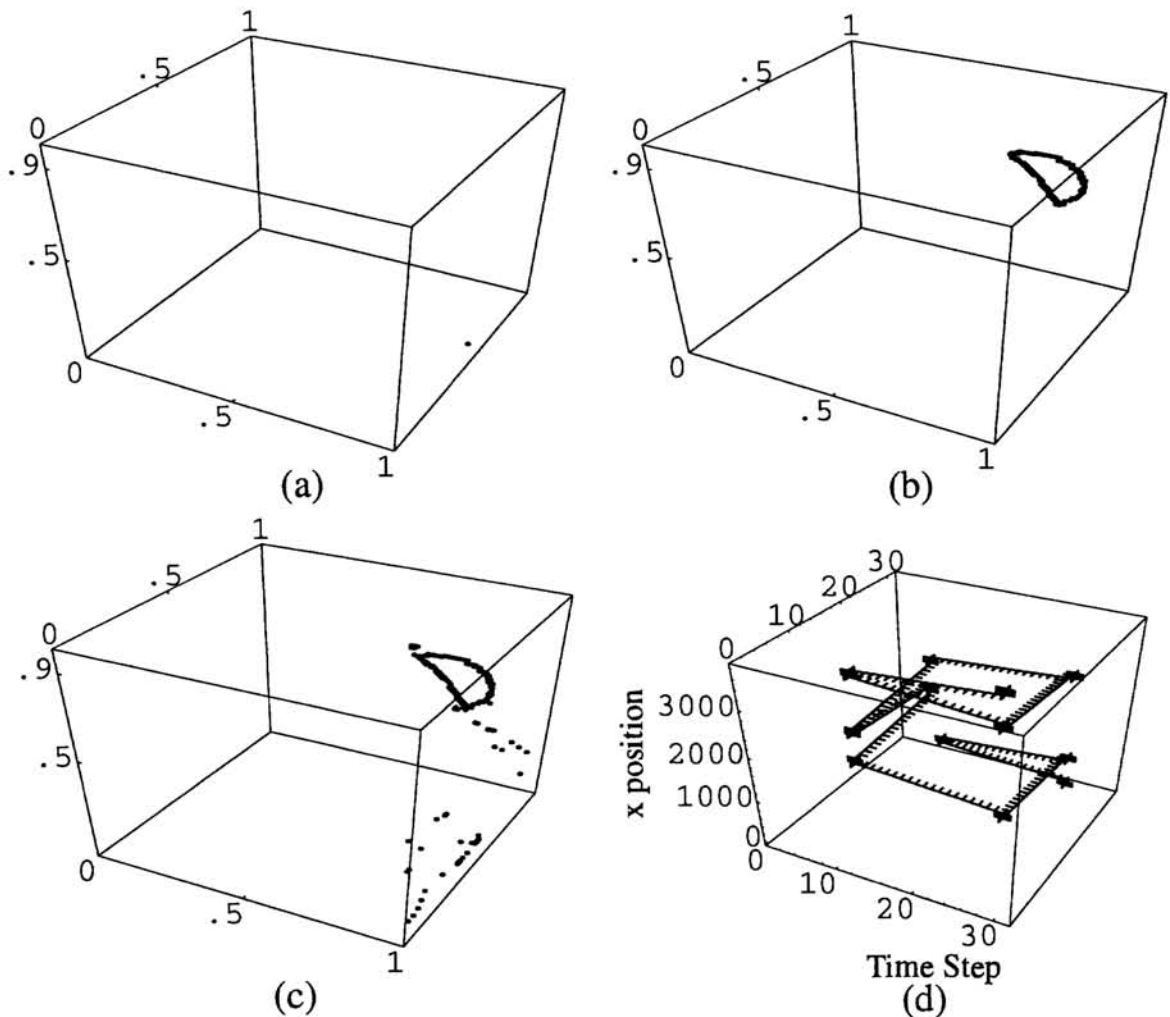

(a)

(b)

(c)

(d)

Figure 5: Limit behavior of the network of the second run. Graphs show the state of the output units Move, Right, Left. (a) Fixed point attractor that results for sequence of 500 "food" signals; (b) Limit cycle attractor that results when a sequence of 500 "no food" signals is given to network; (c) All states visited while traversing the trail; (d) The x position of the ant over time when run on an empty grid.

Chen, D., Giles, C., Sun, G., Chen, H., Less, Y., and Goudreau, M. (1993). Constructive learning of recurrent neural networks. *IEEE International Conference on Neural Networks*, 3:1196–1201.

Cun, Y.L., Denker, J., and Solla, S. (1990). Optimal brain damage. In Touretzky, D., editor, *Advances in Neural Information Processing Systems 2*. Morgan Kaufmann.

Fahlman, S. and Lebiere, C. (1990). The cascade-correlation architecture. In Touretzky, D. S., editor, *Advances in Neural Information Processing Structures 2*, pages 524–532. Morgan Kaufmann.

Fahlman, S. (1991). The recurrent cascade-correlation architecture. In Lippmann, R.,

Moody, J., and Touretzky, D., editors, *Advances in Neural Information Processing Systems 3*, pages 190–196. Morgan Kaufmann, San Mateo.

Fogel, D. (1992). *Evolving Artificial Intelligence.* Ph.D. thesis, University of California, San Diego.

Fogel, D., Fogel, L., and Porto, V. W. (1990). Evolving neural networks. *Biological Cybernetics.* 63:487–493.

Hanson, S. J. (1990). Meiosis networks. In Touretzky, D., editor, *Advances in Neural Information Processing Systems 2*, pages 533–541. Morgan Kaufmann, San Mateo.

Hassibi, B. and Stork, D. G. (1993). Second order derivatives for network pruning: Optimal brain surgeon. In Hanson, S. J., Cowan, J. D., and Giles, C. L., editors, *Advances in Neural Information Processing Systems 5*, pages 164–171. Morgan Kaufmann.

Holland, J. (1975). *Adaptation in Natural and Artificial Systems.* The University of Michigan Press, Ann Arbor, MI.

Jefferson, D., Collins, R., Cooper, C., Dyer, M., Flowers, M., Korf, R., Taylor, C., and Wang, A. (1991). Evolution as a theme in artificial life: The genesys/tracker system. In Langton, C. G., Taylor, C., Farmer, J. D., and Rasmussen, S., editors, *Artificial Life II: Proceedings of the Workshop on Artificial Life.* pages 549–577. Addison-Wesley.

Karunanithi, N., Das, R., and Whitley, D. (1992). Genetic cascade learning for neural networks. In *Proceedings of COGANN-92 International Workshop on Combinations of Genetic Algorithms and Neural Networks.*

Kirkpatrick, S., Gelatt, C. D., and Vecchi, M. P. (1983). Optimization by simulated annealing. *Science*, 220:671-680.

Koza, J. (1992). Genetic evolution and co-evolution of computer programs. In Christopher G. Langton, Charles Taylor, J. D. F. and Rasmussen, S., editors, *Artificial Life II.* Addison Wesley Publishing Company, Reading Mass.

Mozer, M. and Smolensky, P. (1989). Skeletonization: A technique for trimming the fat from a network via relevance assessment. In Touretzky, D., editor, *Advances in Neural Information Processing Systems 1*, pages 107–115. Morgan Kaufmann, San Mateo.

Omlin, C. W. and Giles, C. L. (April 1993). Pruning recurrent neural networks for improved generalization performance. Technical Report Tech Report No 93-6, Computer Science Department, Rensselaer Polytechnic Institute.

Pollack, J. B. (1991). The induction of dynamical recognizer. *Machine Learning.* 7:227-252.

Potter, M. A. (1992). A genetic cascade-correlation learning algorithm. In *Proceedings of COGANN-92 International Workshop on Combinations of Genetic Algorithms and Neural Networks.*

Schaffer, J. D., Whitley, D., and Eshelman, L. J. (1992). Combinations of genetic algorithms and neural networks: A survey of the state of the art. In *Proceedings of COGANN-92 International Workshop on Combinations of Genetic Algorithms and Neural Networks.*